# Efficient Resources Allocation
# for Markov Decision Processes

**Rémi Munos**
CMAP, Ecole Polytechnique, 91128 Palaiseau, France
http://www.cmap.polytechnique.fr/~munos
remi.munos@polytechnique.fr

## Abstract

It is desirable that a complex decision-making problem in an uncertain world be adequately modeled by a Markov Decision Process (MDP) whose structural representation is adaptively designed by a *parsimonious* resources allocation process. Resources include time and cost of exploration, amount of memory and computational time allowed for the policy or value function representation. Concerned about making the best use of the available resources, we address the problem of efficiently estimating where adding extra resources is highly needed in order to improve the expected performance of the resulting policy. Possible application in reinforcement learning (RL), when real-world exploration is highly costly, concerns the detection of those areas of the state-space that need primarily to be explored in order to improve the policy. Another application concerns approximation of continuous state-space stochastic control problems using adaptive discretization techniques for which highly efficient grid points allocation is mandatory to survive high dimensionality. Maybe surprisingly these two problems can be formulated under a common framework: for a given resource allocation, which defines a belief state over possible MDPs, find where adding new resources (thus decreasing the uncertainty of some parameters -transition probabilities or rewards) will most likely increase the expected performance of the new policy. To do so, we use sampling techniques for estimating the contribution of each parameter's probability distribution function (*pdf*) to the expected loss of using an approximate policy (such as the optimal policy of the most probable MDP) instead of the true (but unknown) policy.

## Introduction

Assume that we model a complex decision-making problem under uncertainty by a finite MDP. Because of the limited resources used, the parameters of the MDP (transition probabilities and rewards) are uncertain: we assume that we only know a belief state over their possible values. If we select the most probable values of the parameters, we can build a MDP and solve it to deduce the corresponding optimal policy. However, because of the uncertainty over the true parameters, this policy may not be the one that maximizes the expected cumulative rewards of the

true (but partially unknown) decision-making problem. We can nevertheless use sampling techniques to estimate the expected loss of using this policy. Furthermore, if we assume independence of the parameters (considered as random variables), we are able to derive the contribution of the uncertainty over each parameter to this expected loss. As a consequence, we can predict where adding new resources (thus decreasing the uncertainty over some parameters) will decrease mostly this loss, thus improving the MDP model of the decision-making problem so as to optimize the expected future rewards.

As possible application, in model-free RL we may wish to minimize the amount of real-world exploration (because each experiment is highly costly). Following [1] we can maintain a Dirichlet *pdf* over the transition probabilities of the corresponding MDP. Then, our algorithm is able to predict in which parts of the state space we should make new experiments, thus decreasing the uncertainty over some parameters (the posterior distribution being less uncertain than the prior) in order to optimize the expected payoff.

Another application concerns the approximation of continuous (or large discrete) state-space control problems using variable resolution grids, that requires an efficient resource allocation process in order to survive the "curse of dimensionality" in high dimensions. For a given grid, because of the interpolation process, the approximate back-up operator introduces a local interpolation error (see [4]) that may be considered as a random variable (for example in the random grids of [6]). The algorithm introduced in this paper allows to estimate where we should add new grid-points, thus decreasing the uncertainty over the local interpolation error, in order to increase the expected performance of the new grid representation. The main tool developed here is the calculation of the partial derivative of useful global measures (the value function or the loss of using a sub-optimal policy) with respect to each parameter (probabilities and rewards) of a MDP.

## 1 Description of the problem

We consider a MDP with a finite state-space $X$ and action-space $A$. A transition from a state $x$, action $a$ to a next state $y$ occurs with probability $p(y|x,a)$ and the corresponding (deterministic) reward is $r(x,a)$. We introduce the back-up operator $\mathbf{T}_a$ defined, for any function $W : X \to I\!R$, as

$$\mathbf{T}_a W(x) = \gamma \sum_y p(y|x,a) W(y) + r(x,a) \tag{1}$$

(with some discount factor $0 < \gamma < 1$). It is a contraction mapping, thus the dynamic programming (DP) equation $V(x) = \max_{a \in A} \mathbf{T}_a V(x)$ has a unique fixed point $V$ called the *value function*. Let us define the *Q-values* $Q(x,a) = \mathbf{T}_a V(x)$. The optimal policy $\pi^*$ is the mapping from any state $x$ to the action $\pi^*(x)$ that maximizes the Q-values: $\pi^*(x) = \max_{a \in A} Q(x,a)$.

The **parameters** of the MDP – the probability and the reward functions – are not perfectly known: all we know is a *pdf* over their possible values. This uncertainty comes from the limited amount of allocated resources for estimating those parameters.

Let us choose a specific policy $\hat{\pi}$ (for example the optimal policy of the MDP with the most probable parameters). We can estimate the expected loss of using $\hat{\pi}$ instead of the true (but unknown) optimal policy $\pi^*$. Let us write $\mu = \{\mu_j\}$ the set of all parameters ($p$ and $r$ functions) of a MDP. We assume that we know a probability distribution function $pdf(\mu_j)$ over their possible values. For a MDP $M^\mu$ defined

by its parameters $\mu$, we write $p^\mu(y|x,a)$, $r^\mu(x,a)$, $V^\mu$, $Q^\mu$, and $\pi^\mu$ respectively its transition probabilities, rewards, value function, Q-values, and optimal policy.

## 1.1 Direct gain optimization

We define the **gain** $J^\mu(x;\pi)$ in the MDP $M^\mu$ as the expected sum of discounted rewards obtained starting from state $x$ and using policy $\pi$:

$$J^\mu(x;\pi) = \mathbf{E}[\sum_k \gamma^k r^\mu(x_k, \pi(x_k))|x_0 = x; \pi] \qquad (2)$$

where the expectation is taken for sequences of states $x_k \rightarrow x_{k+1}$ occurring with probability $p^\mu(x_{k+1}|x_k, \pi^\mu(x_k))$. By definition, the **optimal gain** in $M^\mu$ is $V^\mu(x) = J^\mu(x;\pi^\mu)$ which is obtained for the optimal policy $\pi^\mu$. Let $\widehat{V^\mu}(x) = J^\mu(x;\widehat\pi)$ be the **approximate gain** obtained for some approximate policy $\widehat\pi$ in the *same* MDP $M^\mu$. We define the **loss to occur** $L^\mu(x)$ from $x$ when one uses the approximate policy $\widehat\pi$ instead of the optimal one $\pi^\mu$ in $M^\mu$:

$$L^\mu(x) = V^\mu(x) - \widehat{V^\mu}(x) \qquad (3)$$

An example of approximate policy $\widehat\pi$ would be the optimal policy of the most probable MDP, defined by the most probable parameters $\widehat p(y|x,a)$ and $\widehat r(x,a)$.

We also consider the problem of maximizing the global gain from a set of initial states chosen according to some probability distribution $P(x)$. Accordingly, we define the **global gain** of a policy $\pi$: $J^\mu(\pi) = \sum_x J^\mu(x;\pi)P(x)$ and the **global loss** $L^\mu$ of using some approximate policy $\widehat\pi$ instead of the optimal one $\pi^\mu$

$$L^\mu = J^\mu(\pi^\mu) - J^\mu(\widehat\pi) = \sum_x L^\mu(x)P(x) \qquad (4)$$

Thus, knowing the *pdf* over all parameters $\mu$ we can define the **expected global loss** $L = \mathbf{E}_\mu[L^\mu]$.

Next, we would like to define what is the contribution of each parameter uncertainty to this loss, so we know where we should add new resources (thus reducing some parameters uncertainty) in order to decrease the expected global loss. We would like to estimate, for each parameter $\mu_j$,

$$\mathbf{E}[\delta L \,|\, \text{Add } \delta u \text{ units of resource for } \mu_j] \qquad (5)$$

## 1.2 Partial derivative of the loss

In order to quantify (5) we need to be more explicit about the *pdf* over $\mu$. First, we assume the independence of the parameters $\mu_j$ (considered as random variables). Suppose that $pdf(\mu_j) = \mathcal{N}(0, \sigma_j)$ (normal distribution of mean 0 and standard deviation $\sigma_j$). We would like to estimate the variation $\delta L$ of the expected loss $L$ when we make a small change of the uncertainty over $\mu_j$ (consequence of adding new resources), for example when changing the standard deviation of $\delta\sigma_j$ in $pdf(\mu_j)$. At the limit of an infinitesimal variation we obtain the partial derivative $\frac{\partial L}{\partial \sigma_j}$, which when computed for all parameters $\mu_j$, provides the respective contributions of each parameter's uncertainty to the global loss.

Another example is when the $pdf(\mu_j)$ is a uniform distribution of support $[-b_j, b_j]$. Then the partial contribution of $\mu_j$'s uncertainty to the global loss can be expressed as $\frac{\partial L}{\partial b_j}$. More generally, we can define a finite number of characteristic scalar measurements of the *pdf* uncertainty (for example the entropy or the moments) and

compute the partial derivative of the expected global loss with respect to these coefficients. Finally, knowing the actual resources needed to estimate a parameter $\mu_j$ with some uncertainty defined by $pdf(\mu_j)$, we are able to estimate (5).

## 1.3 Unbiased estimator

We sample $N$ sets of parameters $\{\mu^i\}_{i=1..N}$ from the $pdf(\mu)$, which define $N$ MDPs $M^i$. For convenience, we use the superscript $^i$ to refer to the $i$-th MDP sample and the subscript $_j$ for the $j$-th parameter of a variable. We solve each MDP using standard DP techniques (see [5]). This expensive computation that can be speed-up in two ways: first, by using the value function and policy computed for the first MDP as initial values for the other MDPs; second, since all MDPs have the same structure, by computing once for all an efficient ordering (using a topological sort, possibly with loops) of the states that will be used for value iteration.

For each MDP, we compute the global loss $L^i$ of using the policy $\hat{\pi}$ and estimate the expected global loss: $L \simeq \frac{1}{N} \sum_{i=1}^{N} L^i$. In order to estimate the contribution of a parameter's uncertainty to $L$, we derive the partial derivative of $L$ with respect to the characteristic coefficients of $pdf(\mu_j)$. In the case of a reward parameter $\mu_j$ that follows a normal distribution $\mathcal{N}(0, \sigma_j)$, we can write $\mu_j = \sigma_j \xi_j$ where $\xi_j$ follows $\mathcal{N}(0,1)$. The partial derivative of the expected loss $L$ with respect to $\sigma_j$ is

$$\frac{\partial L}{\partial \sigma_j} = \frac{\partial}{\partial \sigma_j} \mathbf{E}_{\mu \sim \mathcal{N}(0,\sigma)}[L^{\mu}] = \frac{\partial}{\partial \sigma_j} \mathbf{E}_{\xi \sim \mathcal{N}(0,1)}[L^{\sigma\xi}] = \mathbf{E}_{\xi \sim \mathcal{N}(0,1)}[\frac{\partial L^{\sigma\xi}}{\partial \mu_j} \xi_j] \quad (6)$$

from which we deduce the unbiased estimator

$$\frac{\partial L}{\partial \sigma_j} \simeq \frac{1}{N} \sum_{i=1}^{N} \frac{\partial L^i}{\partial \mu_j} \frac{\mu_j^i}{\sigma_j} \quad (7)$$

where $\frac{\partial L^i}{\partial \mu_j}$ is the partial derivative of the global loss $L^i$ of MDP $M^i$ with respect to the parameter $\mu_j$ (considered as a variable). For other distributions, we can define similar results to (6) and deduce analogous estimators (for uniform distributions, we have the same estimator with $b_j$ instead of $\sigma_j$).

The remainder of the paper is organized as follow. Section 2 introduces useful tools to derive the partial contribution of each parameter –transition probability and reward– to the value function in a Markov Chain, Section 3 establishes the partial contribution of each parameter to the global loss, allowing to calculate the estimator (7), and Section 4 provides an efficient algorithm. All proofs are given in the full length paper [2].

## 2   Non-local dependencies

### 2.1   Influence of a markov chain

In [3] we introduced the notion of influence of a Markov Chain as a way to measure value function/rewards correlations between states. Let us consider a set of values $V$ satisfying a Bellman equation

$$V(x) = \gamma \sum_{y} p(y|x) V(y) + r(x) \quad (8)$$

We define the discounted cumulative k-chained transition probabilities $p_k(y|x)$:

$$
\begin{aligned}
p_0(y|x) &= \mathbb{1}_{x=y} \quad (=1 \text{ (if } x = y) \text{ or } 0 \text{ (if } x \neq y)) \\
p_1(y|x) &= \gamma \, p(y|x)
\end{aligned}
$$

$$p_2(y|x) = \sum_w p_1(y|w)\, p_1(w|x)$$

$$\dots$$

$$p_k(y|x) = \sum_w p_1(y|w)\, p_{k-1}(w|x)$$

The **influence** $I(y|x)$ of a state $y$ on another state $x$ is defined as $I(y|x) = \sum_{k=0}^{\infty} p_k(y|x)$. Intuitively, $I(y|x)$ measures the expected discounted number of visits of state $y$ starting from $x$; it is also the partial derivative of the value function $V(x)$ with respect to the reward $r(y)$. Indeed $V(x)$ can be expressed as a linear combination of the rewards at $y$ weighted by the influence $I(y|x)$

$$V(x) = \sum_y I(y|x) r(y) \tag{9}$$

We can also define the influence of a state $y$ on a function $f$: $I(y|f(\cdot)) = \sum_x I(y|x) f(x)$ and the influence of a function $f$ on another function $g$ : $I(f(\cdot)|g(\cdot)) = \sum_y I(y|g(\cdot)) f(y)$. In [3], we showed that the influence satisfies

$$I(y|x) = \gamma \sum_w p(y|w) I(w|x) + \mathbb{1}_{x=y} \tag{10}$$

which is a fixed-point equation of a contractant operator (in $1-$norm) thus has a unique solution −the influence− that can be computed by successive iterations. Similarly, the influence $I(y|f(\cdot))$ can be obtained as limit of the iterations

$$I(y|f(\cdot)) \leftarrow \gamma \sum_w p(y|w) I(w|f(\cdot)) + f(y)$$

Thus the computation of the influence $I(y|f(\cdot))$ is cheap (equivalent to solving a Markov chain).

## 2.2 Total derivative of $V$

We wish to express the contribution of all parameters – transition probabilities $p$ and rewards $r$ – (considered as variables) to the value function $V$ by defining the total derivative of $V$ as a function of those parameters. We recall that the total derivative of a function $f$ of several variables $x_1, .., x_n$ is $df = \frac{\partial f}{\partial x_1} dx_1 + ... + \frac{\partial f}{\partial x_n} dx_n$. We already know that the partial derivative of $V(x)$ with respect to the reward $r(z)$ is the influence $I(z|x) = \frac{\partial V(x)}{\partial r(z)}$. Now, the dependency with respect to the transition probabilities has to be expressed more carefully because the probabilities $p(w|z)$ for a given $z$ are dependent (they sum to one). A way to express that is provided in the theorem that follows whose proof is in [2].

**Theorem 1** *For a given state $z$, let us alter the probabilities $p(w|z)$, for all $w$, with some $\delta p(w|z)$ value, such that $\sum_w \delta p(w|z) = 0$. Then $V(x)$ is altered by $\delta V(x) = I(z|x)[\gamma \sum_w V(w) \delta p(w|z)]$. We deduce the total derivative of $V$:*

$$dV(x) = \sum_z I(z|x)[\gamma \sum_w V(w) dp(w|z) + dr(z)]$$

*under the constraint $\sum_w dp(w|z) = 0$ for all $z$.*

## 3 Total derivative of the loss

For a given MDP $M$ with parameters $\mu$ (for notation simplification we do not write the $\mu$ superscript in what follows), we want to estimate the loss of using an approximate policy $\widehat{\pi}$ instead of the optimal one $\pi$. First, we define the **one-step**

**loss** $l(x)$ at a state $x$ as the difference between the gain obtained by choosing the best action $\pi(x)$ then using the optimal policy $\pi$ and the gain obtained by choosing action $\widehat{\pi}(x)$ then the same optimal policy $\pi$

$$l(x) = Q(x, \pi(x)) - Q(x, \widehat{\pi}(x)) \tag{11}$$

Now we consider the **loss** $L(x)$, defined by (3), for an initial state $x$ when we use the approximate policy $\widehat{\pi}$. We can prove that $L(x)$ is the expected discounted cumulative one-step losses $l(x_k)$ for reachable states $x_k$:

$$L(x) = \mathrm{E}[\sum_k \gamma^k \, l(x_k) | x_0 = x; \widehat{\pi}]$$

with the expectation taken in the same sense as in (2).

## 3.1   Decomposition of the one-step loss

We use (9) to decompose the Q-values

$$Q(x, a) = \gamma \sum_w p(w|x, a) \sum_y I(y|w) r(y, \pi(y)) + r(x, a)$$

$$= r(x, a) + \sum_y q(y|x, a) r(y, \pi(y))$$

using the partial contributions $q(y|x, a) = \gamma \sum_w p(w|x, a) I(y|w)$ where $I(y|w)$ is the influence of $y$ on $w$ in the Markov chain derived from the MDP $M$ by choosing policy $\pi$. Similarly, we decompose the one-step loss

$$\begin{aligned}
l(x) &= Q(x, \pi(x)) - Q(x, \widehat{\pi}(x)) \\
&= r(x, \pi(x)) - r(x, \widehat{\pi}(x)) + \sum_y [q(y|x, \pi(x)) - q(y|x, \widehat{\pi}(x))] \, r(y, \pi(y)) \\
&= r(x, \pi(x)) - r(x, \widehat{\pi}(x)) + \sum_y l(y|x) r(y, \pi(y))
\end{aligned}$$

as a function of the partial contributions $l(y|x) = q(y|x, \pi(x)) - q(y|x, \widehat{\pi}(x))$ (see figure 1).

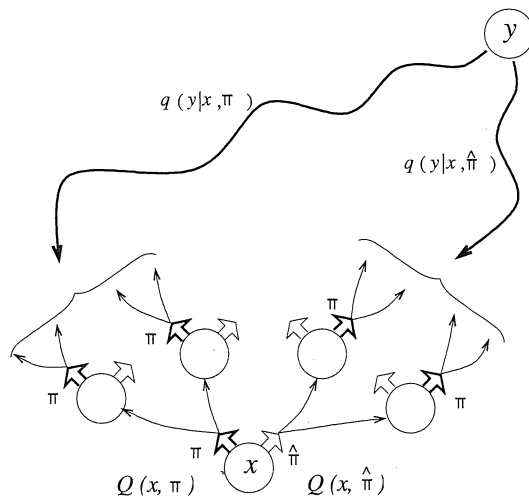

Figure 1: The reward $r(y, \pi(y))$ at state $y$ contributes to the one-step loss $l(x) = Q(x, \pi(x)) - Q(x, \widehat{\pi}(x))$ with the proportion $l(y|x) = q(y|x, \pi(x)) - q(y|x, \widehat{\pi}(x))$.

## 3.2 Total derivative of the one-step loss and global loss

Similarly to section (2.2), we wish to express the contribution of all parameters – transition probabilities $p$ and rewards $r$ – (considered as variables) to the one-step loss function by defining the total derivative of $l$ as a function of those parameters.

**Theorem 2** *Let us introduce the (formal) differential back-up operator $\mathbf{dT}_a$ defined, for any function $W : X \to \mathbb{R}$, as*

$$\mathbf{dT}_a W(x) = \gamma \sum_y W(y) dp(y|x,a) + dr(x,a)$$

*(similar to the back-up operator (1) but using $dp$ and $dr$ instead of $p$ and $r$). The total derivative of the one-step loss is*

$$dl(x) = \sum_z l(z|x) \mathbf{dT}_{\pi(z)} V(z) + \mathbf{dT}_{\pi(x)} V(x) - \mathbf{dT}_{\widehat{\pi}(x)} V(x)$$

*under the constraint $\sum_y dp(y|x,a) = 0$ for all $x$ and $a$.*

**Theorem 3** *Let us introduce the one-step-loss back-up operator $\mathbf{S}$ and its formal differential version $\mathbf{dS}$ defined, for any function $W : X \to \mathbb{R}$, as*

$$
\begin{aligned}
\mathbf{S}W(x) &= \gamma \sum_y p(y|x,\pi(x)) W(y) + l(x) \\
\mathbf{dS}W(x) &= \gamma \sum_y dp(y|x,\pi(x)) W(y) + dl(x)
\end{aligned}
$$

*Then, the loss $L(x)$ at $x$ satisfies Bellman's equation $L = \mathbf{S}L$. The total derivative of the loss $L(x)$ and global loss $L$ are, respectively*

$$
\begin{aligned}
dL(x) &= \sum_z I(z|x) \mathbf{dS}L(z) \\
dL &= \sum_z I(z|P(\cdot)) \mathbf{dS}L(z)
\end{aligned}
$$

from which (after regrouping the contribution to each parameter) we deduce the partial derivatives of the global loss with respect to the rewards and transition probabilities

$$
\begin{aligned}
\frac{\partial L}{\partial r(x,a)} &= I(l(x|\cdot)|P(\cdot))\mathbb{1}_{a=\pi(x)} + I(x|P(\cdot))(\mathbb{1}_{a=\pi(x)} - \mathbb{1}_{a=\widehat{\pi}(x)}) \\
\frac{\partial L}{\partial p(y|x,a)} &= \gamma I(x|P(\cdot))L(y)\mathbb{1}_{a=\pi(x)} + \gamma V(y)\frac{\partial L}{\partial r(x,a)}
\end{aligned}
$$

## 4 A fast algorithm

We use the sampling technique introduced in section 1.3. In order to compute the estimator (7) we calculate the partial derivatives $\frac{\partial L^i}{\partial \mu_j}$ based on the result of the previous section, with the following algorithm.

Given the *pdf* over the parameters $\mu$, select a policy $\widehat{\pi}$ (for example the optimal policy of the most probable MDP). For $i = 1..N$, solve each MDP $M^i$ and deduce

its value function $V^i$, Q-values $Q^i$, and optimal policy $\pi^i$. Deduce the one-step loss $l^i(x)$ from (11). Compute the influence $I(x|P(\cdot))$ (which depends on the transition probabilities $p^i$ of $M^i$) and the influence $I(l^i(x|\cdot)|P(\cdot))$ from which we deduce $\frac{\partial L^i}{\partial r^i(x,a)}$. Then calculate $L^i(x)$ by solving Bellman's equation $L^i = \mathbf{S}L^i$ and deduce $\frac{\partial L^i}{\partial p^i(y|x,a)}$. These partial derivatives enable to compute the unbiased estimator (7). The complexity of solving a discounted MDP with $K$ states, each one connected to $M$ next states, is $\mathcal{O}(KM)$, as is the complexity of computing the influences. Thus, the overall complexity of this algorithm is $\mathcal{O}(NKM)$.

## Conclusion

Being able to compute the contribution of each parameter –transition probabilities and rewards– to the value function (theorem 1) and to the loss of the expected rewards to occur if we use an approximate policy (theorem 3) enables us to use sampling techniques to estimate what are the parameters whose uncertainty are the most harmful to the expected gain. A relevant resource allocation process would consider adding new computational resources to reduce uncertainty over the true value of those parameters. In the examples given in the introduction, this would be doing new experiments in model-free RL for defining more precisely the transition probabilities of some relevant states. In discretization techniques for continuous control problems, this would be adding new grid points in order to improve the quality of the interpolation at relevant areas of the state space in order to maximize the expected gain of the new policy. Initial experiments for variable resolution discretization using random grids show improved performance compared to [3].

### Acknowledgments

I am grateful to Andrew Moore, Drew Bagnell and Auton's Lab members for motivating discussions.

## References

[1] Richard Dearden, Nir Friedman, and David Andre. Model based bayesian exploration. *Proceeding of Uncertainty in Artificial Intelligence*, 1999.

[2] Rémi Munos. Decision-making under uncertainty: Efficiently estimating where extra ressources are needed. *Technical report, Ecole Polytechnique*, 2002.

[3] Rémi Munos and Andrew Moore. Influence and variance of a markov chain : Application to adaptive discretizations in optimal control. *Proceedings of the 38th IEEE Conference on Decision and Control*, 1999.

[4] Rémi Munos and Andrew W. Moore. Rates of convergence for variable resolution schemes in optimal control. *International Conference on Machine Learning*, 2000.

[5] Martin L. Puterman. *Markov Decision Processes, Discrete Stochastic Dynamic Programming*. A Wiley-Interscience Publication, 1994.

[6] John Rust. *Using Randomization to Break the Curse of Dimensionality*. Computational Economics. 1997.

